# A Game-Theoretic Approach to Hypergraph Clustering

**Samuel Rota Bulò**      **Marcello Pelillo**
University of Venice, Italy
{srotabul,pelillo}@dsi.unive.it

## Abstract

Hypergraph clustering refers to the process of extracting maximally coherent groups from a set of objects using high-order (rather than pairwise) similarities. Traditional approaches to this problem are based on the idea of partitioning the input data into a user-defined number of classes, thereby obtaining the clusters as a by-product of the partitioning process. In this paper, we provide a radically different perspective to the problem. In contrast to the classical approach, we attempt to provide a meaningful formalization of the very notion of a cluster and we show that game theory offers an attractive and unexplored perspective that serves well our purpose. Specifically, we show that the hypergraph clustering problem can be naturally cast into a non-cooperative multi-player "clustering game", whereby the notion of a cluster is equivalent to a classical game-theoretic equilibrium concept. From the computational viewpoint, we show that the problem of finding the equilibria of our clustering game is equivalent to locally optimizing a polynomial function over the standard simplex, and we provide a discrete-time dynamics to perform this optimization. Experiments are presented which show the superiority of our approach over state-of-the-art hypergraph clustering techniques.

## 1   Introduction

Clustering is the problem of organizing a set of objects into groups, or *clusters*, in a way as to have similar objects grouped together and dissimilar ones assigned to different groups, according to some similarity measure. Unfortunately, there is no universally accepted formal definition of the notion of a cluster, but it is generally agreed that, informally, a cluster should correspond to a set of objects satisfying two conditions: an *internal coherency* condition, which asks that the objects belonging to the cluster have high mutual similarities, and an *external incoherency* condition, which states that the overall cluster internal coherency decreases by adding to it any external object.

Objects similarities are typically expressed as pairwise relations, but in some applications higher-order relations are more appropriate, and approximating them in terms of pairwise interactions can lead to substantial loss of information. Consider for instance the problem of clustering a given set of $d$-dimensional Euclidean points into lines. As every pair of data points trivially defines a line, there does not exist a meaningful pairwise measure of similarity for this problem. However, it makes perfect sense to define similarity measures over triplets of points that indicate how close they are to being collinear. Clearly, this example can be generalized to any problem of model-based point pattern clustering, where the deviation of a set of points from the model provides a measure of their dissimilarity. The problem of clustering objects using high-order similarities is usually referred to as the *hypergraph clustering problem*.

In the machine learning community, there has been increasing interest around this problem. Zien and co-authors [24] propose two approaches called "clique expansion" and "star expansion", respectively. Both approaches transform the similarity hypergraph into an edge-weighted graph, whose edge-weights are a function of the hypergraph's original weights. This way they are able to tackle

the problem with standard pairwise clustering algorithms. Bolla [6] defines a Laplacian matrix for an unweighted hypergraph and establishes a link between the spectral properties of this matrix and the hypergraph's minimum cut. Rodrìguez [16] achieves similar results by transforming the hypergraph into a graph according to "clique expansion" and shows a relationship between the spectral properties of a Laplacian of the resulting matrix and the cost of minimum partitions of the hypergraph. Zhou and co-authors [23] generalize their earlier work on regularization on graphs and define a hypergraph normalized cut criterion for a $k$-partition of the vertices, which can be achieved by finding the second smallest eigenvector of a normalized Laplacian. This approach generalizes the well-known "Normalized cut" pairwise clustering algorithm [19]. Finally, in [2] we find another work based on the idea of applying a spectral graph partitioning algorithm on an edge-weighted graph, which approximates the original (edge-weighted) hypergraph. It is worth noting that the approaches mentioned above are devised for dealing with higher-order relations, but can all be reduced to standard pairwise clustering approaches [1]. A different formulation is introduced in [18], where the clustering problem with higher-order (super-symmetric) similarities is cast into a nonnegative factorization of the closest hyper-stochastic version of the input affinity tensor.

All the afore-mentioned approaches to hypergraph clustering are partition-based. Indeed, clusters are not modeled and sought directly, but they are obtained as a by-product of the partition of the input data into a fixed number of classes. This renders these approaches vulnerable to applications where the number of classes is not known in advance, or where data is affected by clutter elements which do not belong to any cluster (as in figure/ground separation problems). Additionally, by partitioning, clusters are necessarily disjoint sets, although it is in many cases natural to have overlapping clusters, e.g., two intersecting lines have the point in the intersection belonging to both lines.

In this paper, following [14, 20] we offer a radically different perspective to the hypergraph clustering problem. Instead of insisting on the idea of determining a partition of the input data, and hence obtaining the clusters as a by-product of the partitioning process, we reverse the terms of the problem and attempt instead to derive a rigorous formulation of the very notion of a cluster. This allows one, in principle, to deal with more general problems where clusters may overlap and/or outliers may get unassigned. We found that game theory offers a very elegant and general mathematical framework that serves well our purposes. The basic idea behind our approach is that the hypergraph clustering problem can be considered as a multi-player non-cooperative "clustering game". Within this context, the notion of a cluster turns out to be equivalent to a classical equilibrium concept from (evolutionary) game theory, as the latter reflects both the internal and external cluster conditions alluded to before. We also show that there exists a correspondence between these equilibria and the local solutions of a polynomial, linearly-constrained, optimization problem, and provide an algorithm for finding them. Experiments on two standard hypergraph clustering problems show the superiority of the proposed approach over state-of-the-art hypergraph clustering techniques.

## 2 Basic notions from evolutionary game theory

Evolutionary game theory studies models of strategic interactions (called *games*) among large numbers of anonymous agents. A game can be formalized as a triplet $\Gamma = (P, S, \pi)$, where $P = \{1, \ldots, k\}$ is the set of players involved in the game, $S = \{1, \ldots, n\}$ is the set of *pure strategies* (in the terminology of game-theory) available to each player and $\pi : S^k \to \mathbb{R}$ is the *payoff function*, which assigns a payoff to each *strategy profile*, i.e., the (ordered) set of pure strategies played by the individuals. The payoff function $\pi$ is assumed to be invariant to permutations of the strategy profile. It is worth noting that in general games, each player may have its own set of strategies and own payoff function. For a comprehensive introduction to evolutionary game theory we refer to [22].

By undertaking an evolutionary setting we assume to have a large population of non-rational agents, which are randomly matched to play a game $\Gamma = (P, S, \pi)$. Agents are considered non-rational, because each of them initially chooses a strategy from $S$, which will be always played when selected for the game. An agent, who selected strategy $i \in S$, is called *i-strategist*. Evolution in the population takes place, because we assume that there exists a selection mechanism, which, by analogy with a Darwinian process, spreads the fittest strategies in the population to the detriment of the weakest one, which will in turn be driven to extinction. We will see later in this work a formalization of such a selection mechanism.

The state of the population at a given time $t$ can be represented as a $n$-dimensional vector $\mathbf{x}(t)$, where $x_i(t)$ represents the fraction of $i$-strategists in the population at time $t$. The set of all possible states describing a population is given by

$$\Delta = \left\{ \mathbf{x} \in \mathbb{R}^n \; : \; \sum_{i \in S} x_i = 1 \text{ and } x_i \geq 0 \text{ for all } i \in S \right\} \, ,$$

which is called *standard simplex*. In the sequel we will drop the time reference from the population state, where not necessary. Moreover, we denote with $\sigma(\mathbf{x})$ the *support* of $\mathbf{x} \in \Delta$, i.e., the set of strategies still alive in population $\mathbf{x} \in \Delta$: $\sigma(\mathbf{x}) = \{i \in S \; : \; x_i > 0\}$.

If $\mathbf{y}^{(i)} \in \Delta$ is the probability distribution identifying which strategy the $i$th player will adopt if drawn to play the game $\Gamma$, then the average payoff obtained by the agents can be computed as

$$u(\mathbf{y}^{(1)}, \ldots, \mathbf{y}^{(k)}) = \sum_{(s_1, \ldots, s_k) \in S^k} \pi(s_1, \ldots, s_k) \prod_{j=1}^{k} y_{s_j}^{(j)} \, . \tag{1}$$

Note that (1) is invariant to any permutation of the input probability vectors.

Assuming that the agents are randomly and independently drawn from a population $\mathbf{x} \in \Delta$ to play the game $\Gamma$, the population average payoff is given by $u(\mathbf{x}^k)$, where $\mathbf{x}^k$ is a shortcut for $\mathbf{x}, \ldots, \mathbf{x}$ repeated $k$ times. Furthermore, the average payoff that an $i$-strategist obtains in a population $\mathbf{x} \in \Delta$ is given by $u(\mathbf{e}^i, \mathbf{x}^{k-1})$, where $\mathbf{e}^i \in \Delta$ is a vector with $x_i = 1$ and zero elsewhere.

An important notion in game theory is that of equilibrium [22]. A population $\mathbf{x} \in \Delta$ is in equilibrium when the distribution of strategies will not change anymore, which intuitively happens when every individual in the population obtains the same average payoff and no strategy can thus prevail on the other ones. Formally, $\mathbf{x} \in \Delta$ is a *Nash equilibrium* if

$$u(\mathbf{e}^i, \mathbf{x}^{k-1}) \leq u(\mathbf{x}^k), \qquad \text{for all } i \in S \, . \tag{2}$$

In other words, every agent in the population performs at most as well as the population average payoff. Due to the multi-linearity of $u$, a consequence of (2) is that

$$u(\mathbf{e}^i, \mathbf{x}^{k-1}) = u(\mathbf{x}^k), \qquad \text{for all } i \in \sigma(\mathbf{x}) \, , \tag{3}$$

i.e., all the agents that survived the evolution obtain the same average payoff, which coincides with the population average payoff.

A key concept pertaining to evolutionary game theory is that of an evolutionary stable strategy [7, 22]. Such a strategy is robust to evolutionary pressure in an exact sense. Assume that in a population $\mathbf{x} \in \Delta$, a small share $\epsilon$ of mutant agents appears, whose distribution of strategies is $\mathbf{y} \in \Delta$. The resulting postentry population is given by $\mathbf{w}_\epsilon = (1 - \epsilon)\mathbf{x} + \epsilon \mathbf{y}$. Biological intuition suggests that evolutionary forces select against mutant individuals if and only if the average payoff of a mutant agent in the postentry population is lower than that of an individual from the original population, i.e.,

$$u(\mathbf{y}, \mathbf{w}_\epsilon^{k-1}) < u(\mathbf{x}, \mathbf{w}_\epsilon^{k-1}) \, . \tag{4}$$

A population $\mathbf{x} \in \Delta$ is *evolutionary stable* (or an ESS) if inequality (4) holds for any distribution of mutant agents $\mathbf{y} \in \Delta \setminus \{\mathbf{x}\}$, granted the population share of mutants $\epsilon$ is sufficiently small (see, [22] for pairwise contests and [7] for $n$-wise contests).

An alternative, but equivalent, characterization of ESSs involves a leveled notion of evolutionary stable strategies [7]. We say that $\mathbf{x} \in \Delta$ is an *ESS of level $j$* against $\mathbf{y} \in \Delta$, if there exists $j \in \{0, \ldots, k-1\}$ such that both conditions

$$u(\mathbf{y}^{j+1}, \mathbf{x}^{k-j-1}) \quad < \quad u(\mathbf{y}^j, \mathbf{x}^{k-j}) \, , \tag{5}$$

$$u(\mathbf{y}^{i+1}, \mathbf{x}^{k-i-1}) \quad = \quad u(\mathbf{y}^i, \mathbf{x}^{k-i}) \, , \quad \text{for all } 0 \leq i < j \, , \tag{6}$$

are satisfied. Clearly, $\mathbf{x} \in \Delta$ is an ESS if it satisfies a condition of this form for every $\mathbf{y} \in \Delta \setminus \{\mathbf{x}\}$. It is straightforward to see that any ESS is a Nash equilibrium [22, 7]. An ESS, which satisfies conditions (6) with $j$ never more than $J$, will be called an *ESS of level $J$*. Note that for the generic case most of the preceding conditions will be superfluous, i.e., only ESSs of level 0 or 1 are required [7]. Hence, in the sequel, we will consider only ESSs of level 1. It is not difficult to verify that any ESS (of level 1) $\mathbf{x} \in \Delta$ satisfies

$$u(\mathbf{w}_\epsilon^k) < u(\mathbf{x}^k) \, , \tag{7}$$

for all $\mathbf{y} \in \Delta \setminus \{\mathbf{x}\}$ and small enough values of $\epsilon$.

## 3 The hypergraph clustering game

The hypergraph clustering problem can be described by an edge-weighted hypergraph. Formally, an edge-weighted *hypergraph* is a triplet $H = (V, E, s)$, where $V = \{1, \ldots, n\}$ is a finite set of *vertices*, $E \subseteq \mathcal{P}(V) \setminus \{\emptyset\}$ is the set of (hyper-)edges (here, $\mathcal{P}(V)$ is the power set of $V$) and $s : E \to \mathbb{R}$ is a weight function which associates a real value with each edge. Note that negative weights are allowed too. Although hypergraphs may have edges of varying cardinality, we will focus on a particular class of hypergraphs, called $k$-graphs, whose edges have all fixed cardinality $k \geq 2$.

In this paper, we cast the hypergraph clustering problem into a game, called *(hypergraph) clustering game*, which will be played in an evolutionary setting. Clusters are then derived from the analysis of the ESSs of the clustering game. Specifically, given a $k$-graph $H = (V, E, s)$ modeling a hypergraph clustering problem, where $V = \{1, \ldots, n\}$ is the set of objects to cluster and $s$ is the similarity function over the set of objects in $E$, we can build a game involving $k$ players, each of them having the same set of (pure) strategies, namely the set of objects to cluster $V$. Under this setting, a population $\mathbf{x} \in \Delta$ of agents playing a clustering game represents in fact a cluster, where $x_i$ is the probability for object $i$ to be part of it. Indeed, any cluster can be modeled as a probability distribution over the set of objects to cluster. The payoff function of the clustering game is defined in a way as to favour the evolution of agents supporting highly coherent objects. Intuitively, this is accomplished by rewarding the $k$ players in proportion to the similarity that the $k$ played objects have. Hence, assuming $(v_1, \ldots, v_k) \in V^k$ to be the tuple of objects selected by $k$ players, the payoff function can be simply defined as

$$\pi(v_1, \ldots, v_k) = \begin{cases} \frac{1}{k!} s(\{v_1, \ldots, v_k\}) & \text{if } \{v_1, \ldots, v_k\} \in E, \\ 0 & \text{else}, \end{cases} \tag{8}$$

where the term $1/k!$ has been introduced for technical reasons.

Given a population $\mathbf{x} \in \Delta$ playing the clustering game, we have that the average population payoff $u(\mathbf{x}^k)$ measures the cluster's internal coherency as the average similarity of the objects forming the cluster, whereas the average payoff $u(\mathbf{e}^i, \mathbf{x}^{k-1})$ of an agent supporting object $i \in V$ in population $\mathbf{x}$, measures the average similarity of object $i$ with respect to the cluster.

An ESS of a clustering game incorporates the properties of internal coherency and external incoherency of a cluster:

**internal coherency:** since ESSs are Nash equilibria, from (3), it follows that every object contributing to the cluster, i.e., every object in $\sigma(\mathbf{x})$, has the same average similarity with respect to the cluster, which in turn corresponds to the cluster's overall average similarity. Hence, the cluster is internally coherent;

**external incoherency:** from (2), every object external to the cluster, i.e., every object in $V \setminus \sigma(\mathbf{x})$, has an average similarity which does not exceed the cluster's overall average similarity. There may still be cases where the average similarity of an external object is the same as that of an internal object, mining the cluster's external incoherency. However, since $\mathbf{x}$ is an ESS, from (7) we see that whenever we try to extend a cluster with small shares of external objects, the cluster's overall average similarity drops. This guarantees the external incoherency property of a cluster to be also satisfied.

Finally, it is worth noting that this theory generalizes the dominant-sets clustering framework which has recently been introduced in [14]. Indeed, ESSs of pairwise clustering games, i.e. clustering games defined on graphs, correspond to the dominant-set clusters [20, 17]. This is an additional evidence that ESSs are meaningful notions of cluster.

## 4 Evolution towards a cluster

In this section we will show that the ESSs of a clustering game are in one-to-one correspondence with (strict) local solution of a non-linear optimization program. In order to find ESSs, we will also provide a dynamics due to Baum and Eagon, which generalizes the replicator dynamics [22].

Let $H = (V, E, s)$ be a hypergraph clustering problem and $\Gamma = (P, V, \pi)$ be the corresponding clustering game. Consider the following non-linear optimization problem:

$$\text{maximize} \quad f(\mathbf{x}) = \sum_{e \in E} s(e) \prod_{i \in e} x_i, \quad \text{subject to} \quad \mathbf{x} \in \Delta. \tag{9}$$

It is simple to see that any first-order Karush-Kuhn-Tucker (KKT) point $\mathbf{x} \in \Delta$ of program (9) [13] is a Nash equilibrium of $\Gamma$. Indeed, by the KKT conditions there exist $\mu_i \geq 0$, $i \in S$, and $\lambda \in \mathbb{R}$ such that for all $i \in S$,

$$\nabla f(\mathbf{x})_i + \mu_i - \lambda = \frac{1}{k}u(\mathbf{e}^i, \mathbf{x}^{k-1}) + \mu_i - \lambda = 0 \quad \text{and} \quad \mu_i x_i = 0\,,$$

where $\nabla$ is the gradient operator. From this it follows straightforwardly that $u(\mathbf{e}^i, \mathbf{x}^{k-1}) \leq u(\mathbf{x}^k)$ for all $i \in S$. Moreover, it turns out that any strict local maximizer $\mathbf{x} \in \Delta$ of (9) is an ESS of $\Gamma$. Indeed, by definition, a strict local maximizer of this program satisfies $u(\mathbf{z}^k) = f(\mathbf{z}) < f(\mathbf{x}) = u(\mathbf{x}^k)$, for any $\mathbf{z} \in \Delta \setminus \{\mathbf{x}\}$ close enough to $\mathbf{x}$, which is in turn equivalent to (7) for sufficiently small values of $\epsilon$.

The problem of extracting ESSs of our hypergraph clustering game can thus be cast into the problem of finding strict local solutions of (9). We will address this optimization task using a result due to Baum and Eagon [3], who introduced a class of nonlinear transformations in probability domain.

**Theorem 1** (Baum-Eagon). *Let $P(\mathbf{x})$ be a homogeneous polynomial in the variables $x_i$ with non-negative coefficients, and let $\mathbf{x} \in \Delta$. Define the mapping $\mathbf{z} = \mathcal{M}(\mathbf{x})$ as follows:*

$$z_i = x_i \partial_i P(\mathbf{x}) \Big/ \sum_{j=1}^{n} x_j \partial_j P(\mathbf{x}), \qquad i = 1, \dots, n. \tag{10}$$

*Then $P(\mathcal{M}(\mathbf{x})) > P(\mathbf{x})$, unless $\mathcal{M}(\mathbf{x}) = \mathbf{x}$. In other words $\mathcal{M}$ is a growth transformation for the polynomial $P$.*

The Baum-Eagon inequality provides an effective iterative means for maximizing polynomial functions in probability domains, and in fact it has served as the basis for various statistical estimation techniques developed within the theory of probabilistic functions of Markov chains [4]. It was also employed for the solution of relaxation labelling processes [15].

Since $f(\mathbf{x})$ is a homogeneous polynomial in the variables $x_i$, we can use the transformation of Theorem 1 in order to find a local solution $\mathbf{x} \in \Delta$ of (9), which in turn provides us with an ESS of the hypergraph clustering game. By taking the support of $\mathbf{x}$, we have a cluster under our framework. The complexity of finding a cluster is thus $O(\rho|E|)$, where $|E|$ is the number of edges of the hypergraph describing the clustering problem and $\rho$ is the average number of iteration needed to converge. Note that $\rho$ never exceeded $100$ in our experiments.

In order to obtain the clustering, in principle, we have to find the ESSs of the clustering game. This is a non-trivial, although still feasible, task [21], which we leave as a future extension of this work. By now, we adopt a naive *peeling-off strategy* for our cluster extraction procedure. Namely, we iteratively find a cluster and remove it from the set of objects, and we repeat this process on the remaining objects until a desired number of clusters have been extracted. The set of extracted ESSs with this procedure does not technically correspond to the ESSs of the original game, but to ESSs of sub-games of it. The cost of this approximation is that we unfortunately loose (by now) the possibility of having overlapping clusters.

## 5 Experiments

In this section we present two types of experiments. The first one addresses the problem of line clustering, while the second one addresses the problem of illuminant-invariant face clustering. We tested our approach against Clique Averaging algorithm (CAVERAGE), since it was the best performing approach in [2] on the same type of experiments. Specifically, CAVERAGE outperformed Clique Expansion [10] combined with Normalized cuts, Gibson's Algorithm under sum and product model [9], kHMeTiS [11] and Cascading RANSAC [2]. We also compare against Super-symmetric Non-negative Tensor Factorization (SNTF) [18], because it is the only approach, other than ours, which does not approximate the hypergraph to a graph.

Since both CAVERAGE and SNTF, as opposed to our method, require the number of classes $K$ to be specified, we run them with values of $K \in \{K^* - 1, K^*, K^* + 1\}$ among which the optimal one ($K^*$) is present. This allows us to verify the robustness of the approaches under wrong values of $K$, which may occur in general as the optimal number of clusters is not known in advance.

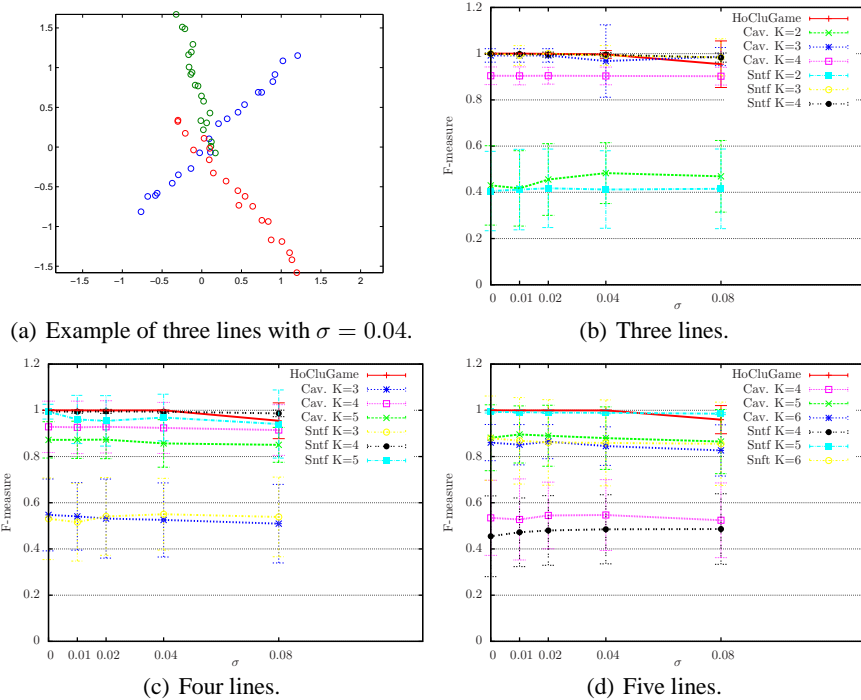

(a) Example of three lines with $\sigma = 0.04$.

(b) Three lines.

(c) Four lines.

(d) Five lines.

Figure 1: Results on clustering $3, 4$ and $5$ lines perturbed with increasing levels of Gaussian noise ($\sigma = 0, 0.01, 0.02, 0.04, 0.08$).

We executed the experiments on a AMD Sempron 3Ghz computer with 1Gb RAM. Moreover, we evaluated the quality of a clustering by computing the average F-measure of each cluster in the ground-truth with the most compatible one in the obtained solution (according to a one-to-one correspondence).

## 5.1 Line clustering

We consider the problem of clustering lines in spaces of dimension greater than two, i.e., given a set of points in $\mathbb{R}^d$, the task is to find sets of collinear points. Pairwise measures of similarity are useless and at least three points are needed. The dissimilarity measure on triplets of points is given by their mean distance to the best fitting line. If $d(i, j, k)$ is the dissimilarity of points $\{i, j, k\}$, the similarity function is given by $s(\{i, j, k\}) = \exp(-d(i, j, k)^2/\sigma^2)$, where $\sigma$ is a scaling parameter, which has been optimally selected for all the approaches according to a small test set.

We conducted two experiments, in order to assess the robustness of the approaches to both local and global noise. Local noise refers to a Gaussian perturbation applied to the points of a line, while global noise consists of random outlier points.

A first experiment consists in clustering $3, 4$ and $5$ lines generated in the 5-dimensional space $[-2, 2]^5$. Each line consists of 20 points, which have been perturbed according to 5 increasing levels of Gaussian noise, namely $\sigma = 0, 0.01, 0.02, 0.04, 0.08$. With this setting there are no outliers and every point should be assigned to a line (e.g., see Figure 1(a)). Figure 1(b) shows the results obtained with three lines. We reported, for each noise level, the mean and the standard deviation of the average F-measures obtained by the algorithms on 30 randomly generated instances. Note that, if the optimal $K$ is used, CAVERAGE and SNTF perform well and the influence of local noise is minimal. This behavior intuitively makes sense under moderate perturbations, because if the approaches correctly partitioned the data without noise, it is unlikely that the result will change by slightly perturbing them. Our approach however achieves good performances as well, although we can notice that with the highest noise level, the performance slightly drops. This is due to the fact that points deviating too much from the overall cluster average collinearity will be excluded as they undermine the cluster's internal coherency. Hence, some perturbed points will be considered outliers. Nevertheless, it is worth noting that by underestimating the optimal number of classes both CAVERAGE and SNTF exhibit a drastic performance drop, whereas the influence of overestimations

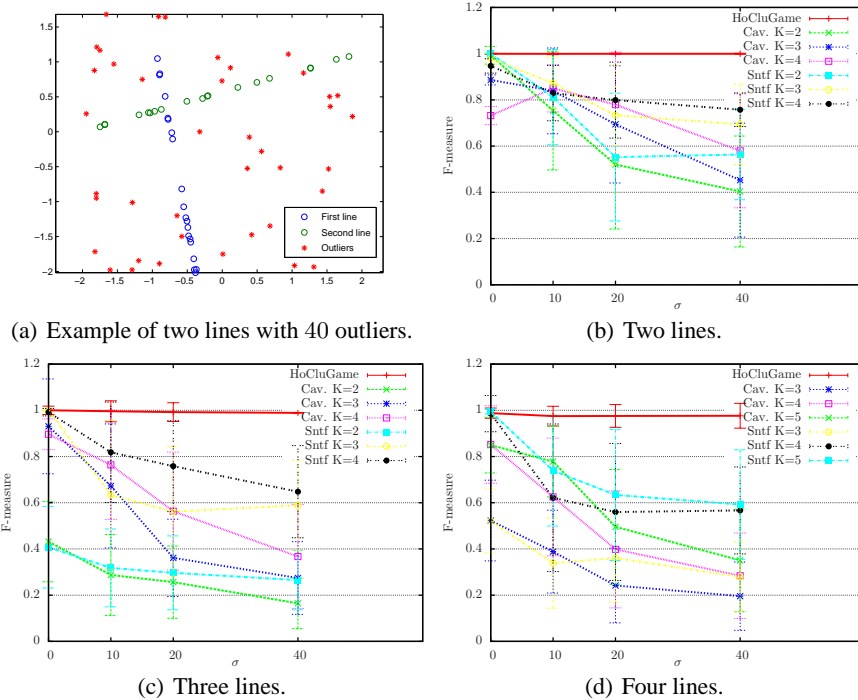

(a) Example of two lines with 40 outliers.

(b) Two lines.

(c) Three lines.

(d) Four lines.

Figure 2: Results on clustering $2, 3$ and $4$ lines with an increasing number of outliers $(0, 10, 20, 40)$.

has a lower impact on the two partition-based algorithms. By increasing the number of lines involved in the experiment from three to four (Figure 1(c)) and to five (Figure 1(d)) the scenario remains almost the same for our approach and SNTF, while we can notice a slight decrease of CAVERAGE's performance.

The second experiment consists in clustering $2, 3$ and $4$ slightly perturbed lines (with fixed local noise $\sigma = 0.01$) generated in the 5-dimensional space $[-2, 2]^5$. Again, each line consists of 20 points. This time however we added also global noise, i.e., $0, 10, 20$ and $40$ random points as outliers (e.g., see Figure 2(a)). Figure 2(b) shows the results obtained with two lines. Here, the supremacy of our approach over partition-based ones is clear. Indeed, our method is not influenced by outliers and therefore it performs almost perfectly, whereas CAVERAGE and SNTF perform well only without outliers and with the optimal $K$. It is interesting to notice that, as outliers are introduced, CAVERAGE and SNTF perform better with $K > 2$. Indeed, the only way to get rid of outliers is to group them in additional clusters. However, since outliers are not mutually similar and intuitively they do not form a cluster, we have that the performance of CAVERAGE and SNTF drop drastically as the number of outliers increases. Finally, by increasing the number of lines from two to three (Figure 2(c)) and to four (Figure 2(d)), the performance of CAVERAGE and SNTF get worse, while our approach still achieves good results.

## 5.2 Illuminant-invariant face clustering

In [5] it has been shown that images of a Lambertian object illuminated by a point light source lie in a three dimensional subspace. According to this result, if we assume that four images of a face form the columns of a matrix then $d = s_4^2/(s_1^2 + \cdots + s_4^2)$ provides us with a measure of dissimilarity, where $s_i$ is the $i$th singular value of this matrix [2]. We use this dissimilarity measure for the face clustering problem and we consider as dataset the Yale Face Database B and its extended version [8, 12]. In total we have faces of 38 individuals, each under 64 different illumination conditions. We compared our approach against CAVERAGE and SNTF on subsets of this face dataset. Specifically, we considered cases where we have faces from 4 and 5 random individuals (10 faces per individual), and with and without outliers. The case with outliers consists in 10 additional faces each from a different individual. For each of those combinations, we created 10 random subsets. Similarly to the case of line clustering, we run CAVERAGE and SNTF with values of $K \in \{K^* - 1, K^*, K^* + 1\}$, where $K^*$ is the optimal one.

| n. of classes: | 4 | | 5 | |
|---|---|---|---|---|
| n. of outliers: | 0 | 10 | 0 | 10 |
| CAVERAGE K=3 | 0.63±0.11 | 0.59±0.07 | - | - |
| CAVERAGE K=4 | **0.96±0.06** | 0.84±0.07 | 0.56±0.14 | 0.58±0.07 |
| CAVERAGE K=5 | 0.91±0.06 | 0.79±0.05 | 0.85±0.12 | 0.83±0.06 |
| CAVERAGE K=6 | - | - | 0.84±0.09 | 0.82±0.06 |
| SNTF K=3 | 0.62±0.12 | 0.58±0.10 | - | - |
| SNTF K=4 | 0.87±0.07 | 0.81±0.08 | 0.61±0.13 | 0.59±0.09 |
| SNTF K=5 | 0.82±0.09 | 0.76±0.09 | 0.86±0.12 | 0.80±0.07 |
| SNTF K=6 | - | - | 0.85±0.08 | 0.79±0.11 |
| HoCluGame | 0.95±0.03 | **0.94±0.02** | **0.95±0.05** | **0.94±0.02** |

Table 1: Experiments on illuminant-invariant face clustering.

In Table 1 we report the average F-measures (mean and standard deviation) obtained by the three approaches. The results are consistent with those obtained in the case of line clustering with the exception of SNTF, which performs worse than the other approaches on this real-world application. CAVERAGE and our algorithm perform comparably well when clustering 4 individuals without outliers. However, our approach turns out to be more robust in every other tested case, i.e., when the number of classes increases and when outliers are introduced. Indeed, CAVERAGE's performance decreases, while our approach yields the same good results.

In both the experiments of line and face clustering the execution times of our approach were higher than those of CAVERAGE, but considerably lower than SNTF. The main reason why CAVERAGE run faster is that our approach and SNTF work directly on the hypergraph without resorting to pairwise relations, which is indeed what CAVERAGE does. Further, we mention that our code was not optimized to improve speed and all the approaches were run without any sampling policy.

## 6 Discussion

In this paper, we offered a game-theoretic perspective to the hypergraph clustering problem. Within our framework the clustering problem is viewed as a multi-player non-cooperative game, and classical equilibrium notions from evolutionary game theory turn out to provide a natural formalization of the notion of a cluster. We showed that the problem of finding these equilibria (clusters) is equivalent to solving a polynomial optimization problem with linear constraints, which we solve using an algorithm based on the Baum-Eagon inequality. An advantage of our approach over traditional techniques is the independence from the number of clusters, which is indeed an intrinsic characteristic of the input data, and the robustness against outliers, which is especially useful when solving figure-ground-like grouping problems. We also mention, as a potential positive feature of the proposed approach, the possibility of finding overlapping clusters (e.g., along the lines presented in [21]), although in this paper we have not explicitly dealt with this problem. The experimental results show the superiority of our approach with respect to the state of the art in terms of quality of solution. We are currently studying alternatives to the plain Baum-Eagon dynamics in order to improve efficiency.

**Acknowledgments.** We acknowledge financial support from the FET programme within EU FP7, under the SIMBAD project (contract 213250). We also thank Sameer Agarwal and Ron Zass for providing us with the code of their algorithms.

## References

[1] S. Agarwal, K. Branson, and S. Belongie. Higher order learning with graphs. In *Int. Conf. on Mach. Learning*, volume 148, pages 17–24, 2006.

[2] S. Agarwal, J. Lim, L. Zelnik-Manor, P. Perona, D. Kriegman, and S. Belongie. Beyond pairwise clustering. In *IEEE Conf. Computer Vision and Patt. Recogn.*, volume 2, pages 838–845, 2005.

[3] L. E. Baum and J. A. Eagon. An inequality with applications to statistical estimation for probabilistic functions of Markov processes and to a model for ecology. *Bull. Amer. Math. Soc.*, 73:360–363, 1967.

[4] L. E. Baum, T. Petrie, G. Soules, and N. Weiss. A maximization technique occurring in the statistical analysis of probabilistic functions of Markov chains. *Ann. Math. Statistics*, 41:164–171, 1970.

[5] P. Belhumeur and D. Kriegman. What is the set of images of an object under all possible lighting conditions. *Int. J. Comput. Vision*, 28(3):245–260, 1998.

[6] M. Bolla. Spectral, euclidean representations and clusterings of hypergraphs. *Discr. Math.*, 117:19–39, 1993.

[7] M. Broom., C. Cannings, and G. T. Vickers. Multi-player matrix games. *Bull. Math. Biology*, 59(5):931–952, 1997.

[8] A. S. Georghiades., P. N. Belhumeur, and D. J. Kriegman. From few to many: illumination cone models for face recognition under variable lighting and pose. *IEEE Trans. Pattern Anal. Machine Intell.*, 23(6):643–660, 2001.

[9] D. Gibson, J. M. Kleinberg, and P. Raghavan. *VLDB*, chapter Clustering categoral data: An approach based on dynamical systems., pages 311–322. Morgan Kaufmann Publishers Inc., 1998.

[10] T. Hu and K. Moerder. Multiterminal flows in hypergraphs. In T. Hu and E. S. Kuh, editors, *VLSI circuit layout: theory and design*, pages 87–93. 1985.

[11] G. Karypis and V. Kumar. Multilevel k-way hypergraph partitioning. *VLSI Design*, 11(3):285–300, 2000.

[12] K. C. Lee, J. Ho, and D. Kriegman. Acquiring linear subspaces for face recognition under variable lighting. *IEEE Trans. Pattern Anal. Machine Intell.*, 27(5):684–698, 2005.

[13] D. G. Luenberger. *Linear and nonlinear programming*. Addison Wesley, 1984.

[14] M. Pavan and M. Pelillo. Dominant sets and pairwise clustering. *IEEE Trans. Pattern Anal. Machine Intell.*, 29(1):167–172, 2007.

[15] M. Pelillo. The dynamics of nonlinear relaxation labeling processes. *J. Math. Imag. and Vision*, 7(4):309–323, 1997.

[16] J. Rodrìguez. On the Laplacian spectrum and walk-regular hypergraphs. *Linear and Multilinear Algebra*, 51:285–297, 2003.

[17] S. Rota Bulò. *A game-theoretic framework for similarity-based data clustering*. PhD thesis, University of Venice, 2009.

[18] A. Shashua, R. Zass, and T. Hazan. Multi-way clustering using super-symmetric non-negative tensor factorization. In *Europ. Conf. on Comp. Vision*, volume 3954, pages 595–608, 2006.

[19] J. Shi and J. Malik. Normalized cuts and image segmentation. *IEEE Trans. Pattern Anal. Machine Intell.*, 22:888–905, 2000.

[20] A. Torsello, S. Rota Bulò, and M. Pelillo. Grouping with asymmetric affinities: a game-theoretic perspective. In *IEEE Conf. Computer Vision and Patt. Recogn.*, pages 292–299, 2006.

[21] A. Torsello, S. Rota Bulò, and M. Pelillo. Beyond partitions: allowing overlapping groups in pairwise clustering. In *Int. Conf. Patt. Recogn.*, 2008.

[22] J. W. Weibull. *Evolutionary game theory*. Cambridge University Press, 1995.

[23] D. Zhou, J. Huang, and B. Schölkopf. Learning with hypergraphs: clustering, classification, embedding. In *Adv. in Neur. Inf. Processing Systems*, volume 19, pages 1601–1608, 2006.

[24] J. Y. Zien, M. D. F. Schlag, and P. K. Chan. Multilevel spectral hypergraph partitioning with arbitrary vertex sizes. *IEEE Trans. on Comp.-Aided Design of Integr. Circ. and Systems*, 18:1389–1399, 1999.

